# Human Reading and the Curse of Dimensionality

**Gale L. Martin**
MCC Austin, TX 78613 galem@mcc.com

## Abstract

Whereas optical character recognition (OCR) systems learn to classify single characters; people learn to classify long character strings in parallel, within a single fixation. This difference is surprising because high dimensionality is associated with poor classification learning. This paper suggests that the human reading system avoids these problems because the number of to-be-classified images is reduced by consistent and optimal eye fixation positions, and by character sequence regularities.

An interesting difference exists between human reading and optical character recognition (OCR) systems. The input/output dimensionality of character classification in human reading is much greater than that for OCR systems (see Figure 1). OCR systems classify one character at time; while the human reading system classifies as many as 8-13 characters per eye fixation (Rayner, 1979) and within a fixation, character category and sequence information is extracted in parallel (Blanchard, McConkie, Zola, and Wolverton, 1984; Reicher, 1969).

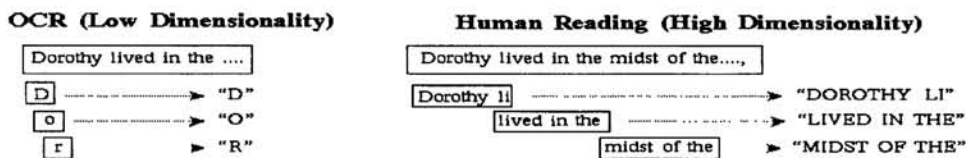

Figure 1: Character classification versus character sequence classification.

This is an interesting difference because high dimensionality is associated with poor classification learning–the so-called *curse of dimensionality* (Denker, et al; 1987; Geman, Bienenstock, & Doursat, 1992). OCR systems are designed to classify single characters to minimize such problems. The fact that most people learn to read quite well even with the high dimensional inputs and outputs, implies that variance

is somehow lowered in this domain, thereby making accurate classification learning possible. The present paper reports on simulations of parallel character classification which suggest that variance is lowered through regularities in eye fixation positions and in character sequences making up valid words.

# 1    Training and Testing Materials

Training and testing materials were drawn from the story *The Wonderful Wizard of Oz* by L. Frank Baum. Images of text lines were created from 120 pages of text (about 160,000 characters, 33,000 total words, or 2,600 different words), which were divided into 6 different font and case conditions of 20 pages each. Three different fonts (variable and constant-width fonts), and two different cases (all upper-case or mixed-case characters) were used. Text line images were normalized with respect to height, but not width. All training and test sets contained an equal mix of the six font/case conditions. Two generalization sets were used, for test and cross-validation, and each consisted of about 14,000 characters.

> Dorothy lived in the midst of the great Kansas Prairies.
> DOROTHY LIVED IN THE MIDST OF THE GREAT KANSAS PRAIRIES.
> Dorothy lived in the midst of the great Kansas Prairies.
> DOROTHY LIVED IN THE MIDST OF THE GREAT KANSAS PRAIRIES.
> Dorothy lived in the midst of the great Kansas Prairies.
> DOROTHY LIVED IN THE MIDST OF THE GREAT KANSAS PRAIRIES.

Figure 2: Samples of the type font and case conditions used in the simulations

# 2    Network Architectures

The simulations used backpropagation networks (Rumelhart, Hinton & Williams, 1986) that extended the local receptive field, shared-weight architecture used in many character-based OCR neural networks (LeCun, et al, 1989; Martin & Pittman, 1991). In the previous single character-based approach, the input to the net is an image of a single character. The output is a vector representing the category of the character. Hidden nodes have local receptive fields that receive input from a spatially local region, (e.g., a 6x6 area) in the preceding layer. Groups of hidden nodes share their weights. Corresponding weights in each receptive field are initialized to the same value and updated by the same value. Different hidden nodes within a group learn to detect the same feature at different locations. A group is depicted as hidden nodes within a single plane of a cube that corresponds to a hidden layer. Different groups occupy different planes in the cube, and learn to detect different features. This architecture biases learning by reducing the number of free parameters available for representing a function. The fact that these nets usually train and generalize well in this domain, and that the local feature detectors that emerge are similar to the oriented-edge and -line detectors found in mammalian visual cortex (Hubel & Wiesel, 1979), suggests that the bias is at least roughly appropriate.

The extension of this character network to a character-sequence network is illustrated in Figure 3, where $n$ (number of to-be-classified characters) is equal to 4. Each output node represents a character category (e.g., "D") in one of the nth ordinal positions (e.g., "First character on the left"). The size of the input window is expanded horizontally to cover at least the $n$ widest characters ("WWWW"). When the character string is made up of relatively narrow characters, more than $n$ characters will appear in the input window and the network must learn to ignore

them. Increasing input/output dimensionality is accomplished by expanding the number of hidden nodes horizontally. Network capacity is described by the depth of each hidden layer (the number of different features detected), as well as by the width of each hidden layer (the spatial coverage of the network).

The network is potentially sensitive to both local and global visual information. Local receptive fields build in a sensitivity to letter features. Shared weights make learning transfer possible across representations of the same character at different positions. Output nodes are globally connected to all the nodes in the second hidden layer, but not with one another or with any word-level representations. Networks were trained until the training set accuracy failed to improve by at least .1% over 5 epochs, or overfitting became evident from periodic testing with the generalization test set.

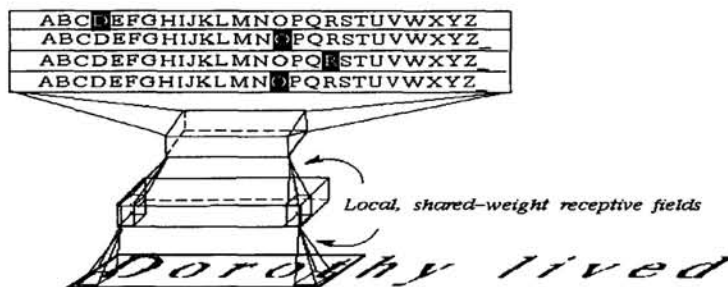

Figure 3: Net architecture for parallel character sequence classification, n=4 chars.

## 3   Effects of Dimensionality on Training Difficulty and Generalization

Experiment 1 provides a baseline measure of the impact of dimensionality. Increases in dimensionality result in exponential increases in the number of input and output patterns and the number of mapping functions. As a result, training problems arise due to limitations in network capacity or search scope. Generalization problems arise because it becomes impractical to use training sets large enough to obtain a good estimate of the underlying function. Four different levels of dimensionality were used (see Figure 4), from an input window of 20x20 pixels, with 1 to-be-classified character to an 80x20 window, with 4 to-be-classified characters ). Input patterns were generated by starting the window at the left edge of the text line such that the first character was centered 10 pixels from the left of the window, and then successively scanning across the text line at each character position. Five training set sizes were used (about 700 samples to 50,000). Two relative network capacities were used (15 and 18 different feature detectors per hidden layer). Forty different

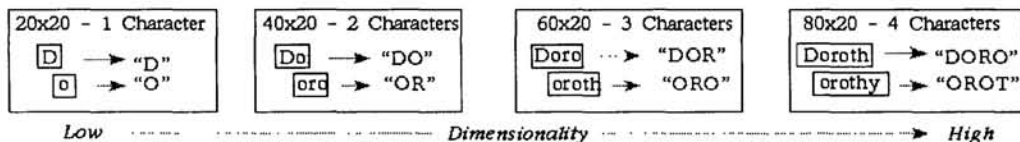

Figure 4: Four levels of input/output dimensionality used in the experiment.

networks were trained, one for each combination of dimensionality, training set size and relative network capacity (4x5x2). Training difficulty is described by asymptotic accuracy achieved on the training set and by amount of training required to reach the asymptote. Generalization is reported for both the test set (used to check for overfitting) and the cross-validation set. The results (see Figure 5) are consistent

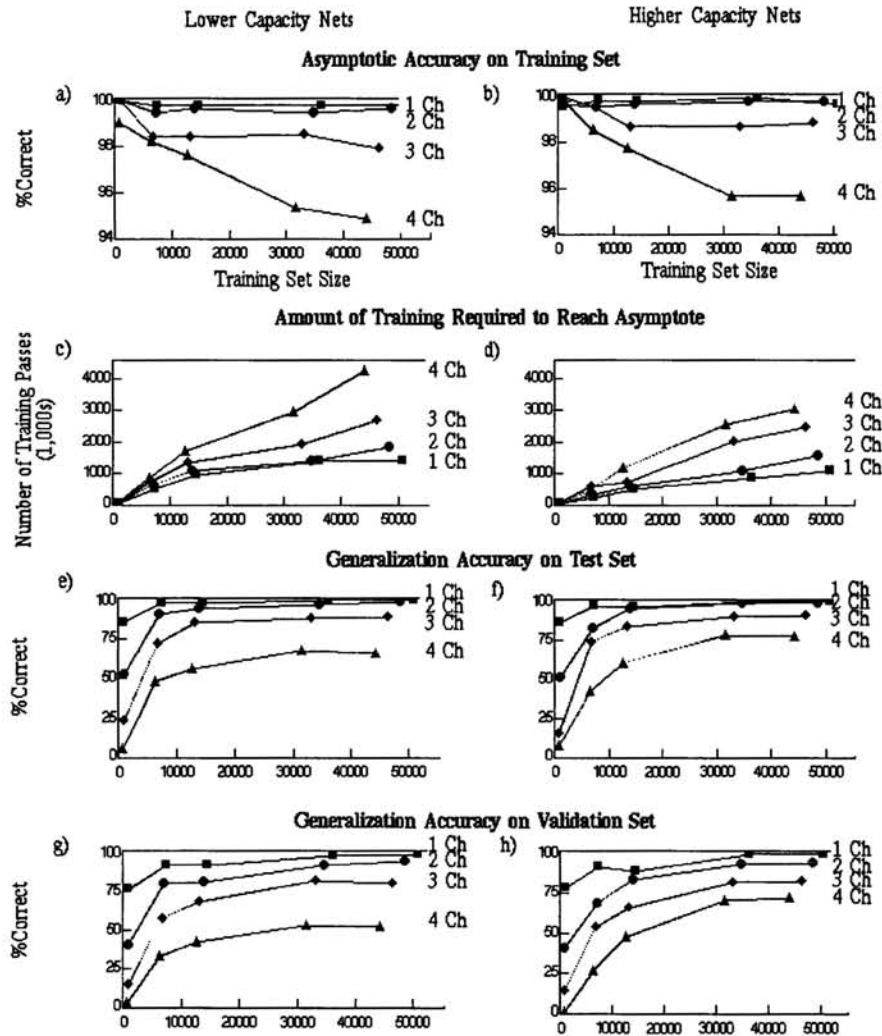

Figure 5: Impact of dimensionality on training and generalization.

with expectations. Increasing dimensionality results in increased training difficulty and lower generalization. Since the problems associated with high dimensionality occur in both training and test sets, and seem to be alleviated somewhat in the high capacity nets, they are presumably due to both capacity/search limitations and insufficient sample size.

## 4   Regularities in Window Positioning

One way human reading might reduce the problems associated with high dimensionality is to constrain eye fixation positions during reading; thereby reducing the number of different input images the system must learn to classify. Eye movement

studies suggest that, although fixation positions within words do vary, there are consistencies Rayner, 1979). Moreover, the particular locations fixated, slightly to the left of the middle of words, appear to be optimal. People are most efficient at recognizing words at these locations (O'Regan & Jacobs, 1992). These fixation positions reduce variance by reducing the average variability in the positions of ordered characters within a word. Position variability increases as a function of distance from the fixated character. The average distance of characters within a word is minimized when the fixation position is toward the center of a word, as compared to when it is at the beginning or end of a word.

Experiment 2 simulated consistent and optimal positioning with an 80x20 input window fixated on the 3rd character. Only words of 3 or more characters were fixated (see Figure 6). The network learned to classify the first 4 characters in the word. This condition was compared to a *consistent positioning only* condition, in which the input window was fixated on the first character of a word. Two control conditions were also examined. They were replications of the 20x20-1Character and the 80x20-4 Character conditions of Experiment 1, except that in the first case, the network was trained and tested only on the first 4 characters in each word and in the second case, the network was trained as before but was tested with the window fixated on the first character of the word. Four levels of training set size were used and three replications of each training set size x window conditions were run (4 x 4 x 3 = 48 networks trained and tested). All networks employed 18 different feature detectors for each hidden layer. The results (see Figure 7) support the idea that

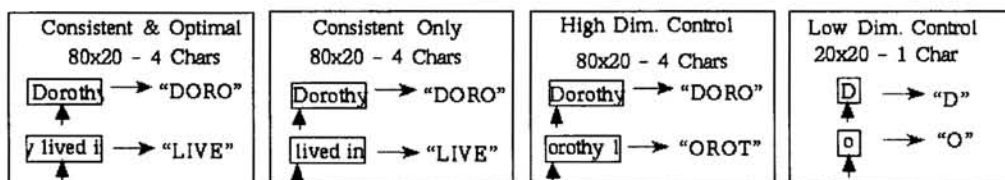

Figure 6: Window positioning and dimensionality manipulations in Experiment 2

consistent and optimal positioning reduces variance, as indicated by reductions in training difficulties and improved generalization. The consistent and optimal positioning networks achieved training and generalization results superior to the high dimensionality control condition, and equivalent to, or better than those for the low dimensionality control. They were also slightly better than the consistent positioning only nets.

## 5 Character Sequence Regularities

Since only certain character sequences are allowed in words, character sequence regularities in words may also reduce the number of distinct images the system must learn to classify. The system may also reduce variance by optimizing accuracy on highest frequency words. These hypotheses were tested by determining whether or not the three consistent and optimal positioning networks trained on the largest training set in Experiment 2, were more accurate in classifying high frequency words, as compared to low frequency words; and more accurate in classifying words as compared to pronounceable non-words or random character strings. The control condition used the networks trained in the low dimensional control (20x20 -1 Character) condition from Experiment 2. Human reading exhibits increased efficiency/accuracy in classifying high frequency as compared to low frequency words

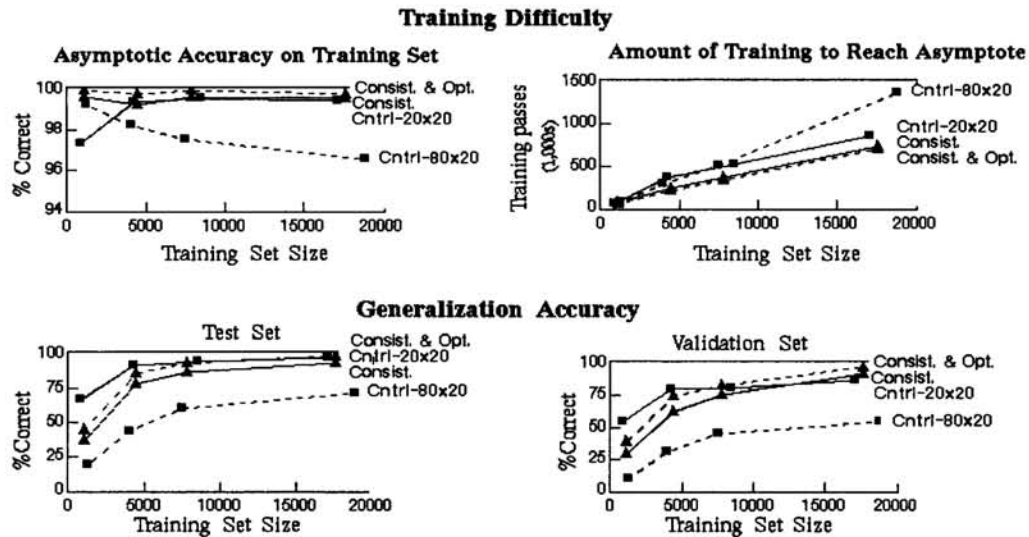

Figure 7: Impact of consistent & optimal window positions.

(Howes & Solomon, 1951; Solomon & Postman, 1952) , and in classifying charac-
ters in words as compared to pronounceable non-words or random character strings
(Baron & Thurston, 1973; Reicher, 1969). Experiment 3 involved creating a list
of 30 4-letter words drawn from the Oz text, of which 15 occurred very frequently
in the text (e.g., SAID), and 15 occurred infrequently (e.g., PAID), and creating
a list of 30 4-letter pronounceable non-words (e.g., TOID) and a list of 30 4-letter
random strings (e.g., SDIA). Each string was reproduced in each of the 6 font /case
conditions and labeled to create a test set. One further condition involved creating a
version of the word list in which the cases of the characters aLtErNaTeD. Psycholo-
gists used this manipulation to demonstrate that the advantages in processing words
can not simply be due to the use of word-shape feature detectors, since the word
advantage carries over to the alternating case condition, which destroys word-level
features (McClelland, 1976).

Consistent with human reading (see Figure 8), the character-sequence-based net-
works were most accurate on high frequency words and least accurate for low fre-
quency words. The character-sequence-based networks also showed a progressive
decline in accuracy as the character string became less word-like. The advantage
for word-like strings can not be due to the use of word shape feature detectors
because accuracy on aLtErNaTiNg case words, where word shape is unfamiliar,
remains quite high.

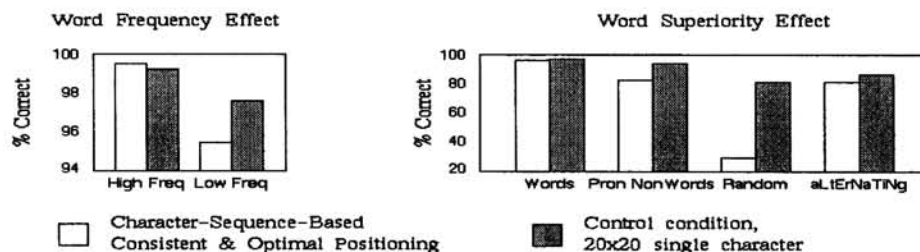

Figure 8: Sensitivity to word frequency and character sequence regularities

The present results raise questions about the role played by high dimensionality in determining reading disabilities and difficulties. Reading difficulties have been associated with reduced perceptual spans (Rayner, 1986; Rayner, et al., 1989), and with irregular eye fixation patterns (Rayner & Pollatsek, 1989). This suggests that some reading difficulties and disorders may be related to problems in generating the precise eye movements necessary to maintain consistent and optimal eye fixations. More generally, these results highlight the importance of considering the role of character classification in learning to read, particularly since content factors, such as word frequency, appear to influence even low-level classification operations.

## References

Blanchard, H., McConkie, G., Zola, D., & Wolverton, G. (1984) Time course of visual information utilization during fixations in reading. *Jour. of Exp. Psych.: Human Perc. & Perf., 10,* 75-89.

Denker, J., Schwartz, D., Wittner, B., Solla, S., Howard, R., Jackel, L., & Hopfield, J. (1987) Large automatic learning, rule extraction and generalization, *Complex Systems, 1,* 877-933.

Geman, S., Bienenstock, E., and Doursat, R. (1992) Neural networks and the bias/variance dilemma. *Neural Computation, 4,* 1-58.

Howes, D. and Solomon, R. L. (1951) Visual duration threshold as a function of word probability. *Journal of Exp. Psych., 41,* 401-410.

Hubel, D. & Wiesel, T. (1979) Brain mechanisms of vision. *Sci. Amer., 241,* 150-162.

LeCun, Y., Boser, B., Denker, J., Henderson, D., Howard, R., Hubbard, W., & Jackel, L. (1990) Handwritten digit recognition with a backpropagation network. In *Adv. in Neural Inf. Proc. Sys. 2,* D. Touretzky (Ed) Morgan Kaufmann.

Martin, G. L. & Pittman, J. A. (1991) Recognizing hand-printed letters and digits using backpropagation learning. *Neural Computation, 3,* 258-267.

McClelland, J. L. (1976) Preliminary letter identification in the perception of words and nonwords. *Jour. of Exp. Psych.: Human Perc. & Perf., 2,* 80-91.

O'Regan, J. & Jacobs, A.(1992) Optimal viewing position effect in word recognition. *Jour. of Exp. Psych.: Human Perc.& Perf., 18,* 185-197.

Rayner, K. (1986) Eye movements and the perceptual span in beginning and skilled readers. *Jour. of Exp. Child Psych., 41,* 211-236.

Rayner, K. (1979) Eye guidance in reading. *Perception, 8,* 21-30.

Rayner, K., Murphy, L., Henderson, J. & Pollatsek, A. (1989) Selective attentional dyslexia. *Cognitive Neuropsych., 6,* 357-378.

Rayner, K. & Pollatsek, A. (1989) *The Psychology of reading.* Prentice Hall

Reicher, G. (1969) Perceptual recognition as a function of meaningfulness of stimulus material. *Jour. of Exp. Psych., 81,* 274-280.

Rumelhart, D., Hinton, G., and Williams, R. (1986) Learning internal representations by error propagation. In D. Rumelhart and J. McClelland, (Eds) *Parallel Distributed Processing, 1.* MIT Press.

Solomon, R. & Postman, L. (1952) Frequency of usage as a determinant of recognition thresholds for words. *Jour. of Exp. Psych., 43,* 195-210.
